# Learning Switching Linear Models of Human Motion

**Vladimir Pavlović and James M. Rehg**
Compaq - Cambridge Research Lab
Cambridge, MA 02139
{*vladimir.pavlovic,jim.rehg*}*@compaq.com*

**John MacCormick**
Compaq - System Research Center
Palo Alto, CA 94301
{*john.maccormick*}*@compaq.com*

## Abstract

The human figure exhibits complex and rich dynamic behavior that is both nonlinear and time-varying. Effective models of human dynamics can be learned from motion capture data using switching linear dynamic system (SLDS) models. We present results for human motion synthesis, classification, and visual tracking using learned SLDS models. Since exact inference in SLDS is intractable, we present three approximate inference algorithms and compare their performance. In particular, a new variational inference algorithm is obtained by casting the SLDS model as a Dynamic Bayesian Network. Classification experiments show the superiority of SLDS over conventional HMM's for our problem domain.

## 1 Introduction

The human figure exhibits complex and rich dynamic behavior. Dynamics are essential to the classification of human motion (e.g. gesture recognition) as well as to the synthesis of realistic figure motion for computer graphics. In visual tracking applications, dynamics can provide a powerful cue in the presence of occlusions and measurement noise. Although the use of kinematic models in figure motion analysis is now commonplace, dynamic models have received relatively little attention. The kinematics of the figure specify its degrees of freedom (e.g. joint angles and torso pose) and define a state space. A stochastic dynamic model imposes additional structure on the state space by specifying a probability distribution over state trajectories.

We are interested in learning dynamic models from motion capture data, which provides a training corpus of observed state space trajectories. Previous work by a number of authors has applied Hidden Markov Models (HMMs) to this problem. More recently, switching linear dynamic system (SLDS) models have been studied in [5, 12]. In SLDS models, the Markov process controls an underlying linear dynamic system, rather than a fixed Gaussian measurement model.[1] By mapping discrete hidden states to piecewise linear measurement models, the SLDS framework has potentially greater descriptive power than an HMM. Offsetting this advantage is the fact that exact inference in SLDS is intractable. Approximate inference algorithms are required, which in turn complicates SLDS learning.

In this paper we present a framework for SLDS learning and apply it to figure motion modeling. We derive three different approximate inference schemes: Viterbi [13], variational, and GPB2 [1]. We apply learned motion models to three tasks: classification, motion synthesis, and visual tracking. Our results include an empirical comparison between SLDS

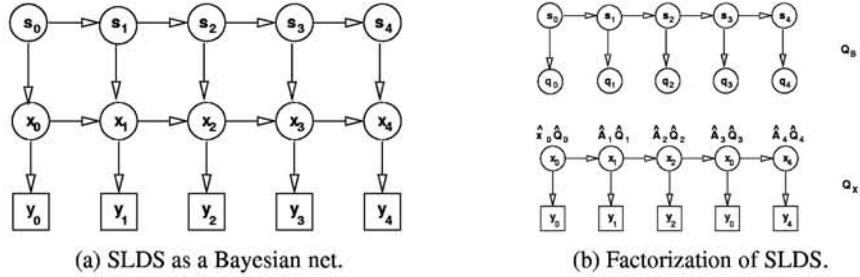

(a) SLDS as a Bayesian net.                    (b) Factorization of SLDS.

Figure 1: (a) SLDS model as Dynamic Bayesian Network. $s$ is discrete switch state, $x$ is continuous state, and $y$ is its observation. (b) Factorization of SLDS into decoupled HMM and LDS.

and HMM models on classification and one-step ahead prediction tasks. The SLDS model class consistently outperforms standard HMMs even on fairly simple motion sequences.

Our results suggest that SLDS models are a promising tool for figure motion analysis, and could play a key role in applications such as gesture recognition, visual surveillance, and computer animation. In addition, this paper provides a summary of approximate inference techniques which is lacking in the previous literature on SLDS. Furthermore, our variational inference algorithm is novel, and it provides another example of the benefit of interpreting classical statistical models as (mixed-state) graphical models.

## 2   Switching Linear Dynamic System Model

A switching linear dynamic system (SLDS) model describes the dynamics of a complex, nonlinear physical process by switching among a set of linear dynamic models over time. The system can be described using the following set of state-space equations,

$$ x_{t+1} = A(s_{t+1})x_t + v_{t+1}(s_{t+1}), \ y_t = Cx_t + w_t, \ Pr(s_{t+1} = i | s_t = j) = \Pi(i, j), $$

for the plant and the switching model. The meaning of the variables is as follows: $x_t \in \Re^N$ denotes the hidden state of the LDS, and $v_t$ is the state noise process. Similarly, $y_t \in \Re^M$ is the observed measurement and $w_t$ is the measurement noise. Parameters $A$ and $C$ are the typical LDS parameters: the state transition matrix and the observation matrix, respectively. We assumed that the LDS models a Gauss-Markov process with i.i.d. Gaussian noise processes $v_t(s_t) \sim \mathcal{N}(0, Q(s_t))$. The switching model is a discrete first order Markov process with state variables $s_t$ from a set of $S$ states. The switching model is defined with the state transition matrix $\Pi$ and an initial state distribution $\pi_0$. The LDS and switching process are coupled due to the dependence of the LDS parameters $A$ and $Q$ on the switching state $s_t$: $A(s_t = i) = A_i$, $Q(s_t = i) = Q_i$.

The complex state space representation is equivalently depicted by the DBN dependency graph in Figure 1(a). The dependency graph implies that the *joint distribution* $P(\mathcal{Y}_T, \mathcal{X}_T, \mathcal{S}_T)$ over the variables of the SLDS can be written as

$$ Pr(s_0) \prod_{t=1}^{T-1} Pr(s_t | s_{t-1}) Pr(x_0 | s_0) \prod_{t=1}^{T-1} Pr(x_t | x_{t-1}, s_t) \prod_{t=0}^{T-1} Pr(y_t | x_t), \qquad (1) $$

where $\mathcal{Y}_T$, $\mathcal{X}_T$, and $\mathcal{S}_T$ denote the sequences (of length $T$) of observations and hidden state variables. From the Gauss-Markov assumption on the LDS and the Markov switching assumption, we can expand Equation 1 into the parameterized joint pdf of the SLDS of duration T.

Learning in complex DBNs can be cast as ML learning in general Bayesian networks. The generalized EM algorithm can then be used to find optimal values of DBN parameters $\{A, C, Q, R, \Pi, \pi_0\}$. Inference, which is addressed in the next section, is the most complex

step in SLDS learning. Given the sufficient statistics from the inference phase, the *parameter update equations* in the maximization (M) step are easily obtained by maximizing the expected log of Equation 1 with respect to the LDS and MC parameters (see [13]).

## 3  Inference in SLDS

The goal of inference in complex DBNs is to estimate the posterior $P(\mathcal{X}_T, \mathcal{S}_T|\mathcal{Y}_T)$. If there were no switching dynamics, the inference would be straightforward – we could infer $\mathcal{X}_T$ from $\mathcal{Y}_T$ using LDS inference. However, the presence of switching dynamics makes exact inference exponentially hard, as the distribution of the system state at time $t$ is a mixture of $S^t$ Gaussians. Tractable, approximate inference algorithms are therefore required. We describe three methods: Viterbi, variational, and generalized Pseudo Bayesian.

### 3.1  Approximate Viterbi Inference

Viterbi approximation approach finds the most likely sequence of switching states $\mathcal{S}_T^*$ for a given observation sequence $\mathcal{Y}_T$. Namely, the desired posterior $P(\mathcal{X}_T, \mathcal{S}_T|\mathcal{Y}_T)$ is approximated by its mode $Pr(\mathcal{X}_T|\mathcal{S}_T^*, \mathcal{Y}_T)$. It is well known how to apply Viterbi inference to discrete state hidden Markov models and continuous state Gauss-Markov models. Here we review an algorithm for approximate Viterbi inference in SLDSs presented in [13].

We have shown in [13] that one can use a recursive procedure to find the best switching sequence $\mathcal{S}_T^* = \arg\max_{\mathcal{S}_T} Pr(\mathcal{S}_T|\mathcal{Y}_T)$. In the heart of this recursion lays the approximation of the partial probability of the swiching sequence and observations up to time $t$, $J_{t,i} = \max_{\mathcal{S}_{t-1}} Pr(\mathcal{S}_{t-1}, s_t = i, \mathcal{Y}_t) \approx$

$$\max_j \left\{ Pr\left(y_t|s_t = i, s_{t-1} = j, \mathcal{S}_{t-2}^*(j), \mathcal{Y}_{t-1}\right) Pr\left(s_t = i|s_{t-1} = j\right) J_{t-1,j}\right\}. \quad (2)$$

The two scaling components are the likelihood associated with the transition $i \rightarrow j$ from $t$ to $t - 1$, and the probability of discrete SLDS switching from $j$ to $i$. They have the notion of a "transition probability" and we denote them together by $J_{t|t-1,i,j}$

The likelihood term can easily be found using Kalman updates, concurrent with the recursion of Equation 2. See [13] for details. The Viterbi inference algorithm can now be written

```
Initialize LDS state estimates x̂_{0|-1,i} and Σ_{0|-1,i};
Initialize J_{0,i};
for t = 1 : T - 1
    for i = 1 : S
        for j = 1 : S
            Predict and filter LDS state estimates x̂_{t|t,i,j} and Σ_{t|t,i,j};
            Find j → i "transition probability" J_{t|t-1,i,j};
        end
        Find best transition ψ_{t-1,i} into state i;
        Update sequence probabilities J_{t,i} and LDS state estimates x̂_{t|t,i} and Σ_{t|t,i};
    end
end
Find "best" final switching state i*_{T-1} and backtrace the best switching sequence S*_T;
Do RTS smoothing for S = S*_T;
```

### 3.2  Approximate Variational Inference

A general structured variational inference technique for Bayesian networks is described in [8]. Namely, an $\eta$-parameterized distribution $Q(\eta)$ is constructed which is "close" to the desired conditional distribution $P$ but is computionally feasible. In our case we define $Q$ by decoupling the switching and LDS portions of SLDS as shown in Figure 1(b). The original distribution is factorized into two independent distributions, a Hidden Markov Model (HMM) $Q_S$ with variational parameters $\{q_0, \ldots, q_{T-1}\}$ and a time-varying LDS $Q_X$ with variational parameters $\{\hat{x}_0, \hat{A}_0, \ldots, \hat{A}_{T-1}, \hat{Q}_0, \ldots, \hat{Q}_{T-1}\}$.

The optimal values of the variational parameters $\eta$ are obtained by minimizing the KL-divergence w.r.t. $\eta$. For example, we arrive at the following optimal variational parameters:

$$\hat{Q}_t^{-1} = \sum_{i=0}^{S-1} Q_i^{-1} Pr(s_t = i) + \sum_{i=0}^{S-1} A_i' Q_i^{-1} A_i Pr(s_{t+1} = i) - \hat{A}_{t+1}' \hat{Q}_{t+1}^{-1} \hat{A}_{t+1}$$

$$\hat{A}_t = \hat{Q}_t \sum_{i=0}^{S-1} Q_i^{-1} A_i Pr(s_t = i)$$

$$\log q_t(i) = -\left\langle (x_t - A_i x_{t-1})' \hat{Q}_i^{-1} (x_t - A_i x_{t-1}) \right\rangle - \log |\hat{Q}_{t,i}| \quad (3)$$

To obtain the terms $Pr(s_t) = Pr(s_t | q_0, \ldots, q_{T-1})$ we use the inference in the HMM with output "probabilities" $q_t$. Similarly, to obtain $\langle x_t \rangle = E[x_t | \mathcal{Y}_T]$ we perform LDS inference in the decoupled time-varying LDS via RTS smoothing. Equation 3 together with the inference solutions in the decoupled models form a set of fixed-point equations. Solution of this fixed-point set is a tractable approximation to the intractable inference of the fully coupled SLDS. The variational inference algorithm for fully coupled SLDSs can now be summarized as:

```
error = ∞;
Initialize Pr(sₜ);
while ( KL divergence > maxError )
        Find Q̂ₜ, Âₜ, x̂₀ from Pr(sₜ) (Eq. 3);
        Estimate ⟨xₜ⟩, ⟨xₜxₜ'⟩ and ⟨xₜxₜ₋₁'⟩ from yₜ using time-varying LDS inference;
        Find qₜ from ⟨xₜ⟩, ⟨xₜxₜ'⟩ and ⟨xₜxₜ₋₁'⟩ (Eq. 3);
        Estimate Pr(sₜ) from qₜ using HMM inference.
end
```

Variational parameters in Equation 3 have intuitive interpretation. LDS parameters $\hat{A}_t$ and $\hat{Q}_t^{-1}$ define the best unimodal representation of the corresponding switching system and are, roughly, averages of original parameters weighted by a best estimates of the switching states $P(s_t)$. HMM variational paremeters $\log q_t$, on the other hand, measure the agreement of each individual LDS with the data.

### 3.3 Approximate Generalized Pseudo Bayesian Inference

The Generalized Psuedo Bayesian [1, 9] (GPB) approximation scheme is based on the general idea of "collapsing" a mixture of $M^t$ Gaussians onto a mixture of $M^r$ Gaussians, where $r < t$ (see [12] for a detailed review). While there are several variations on this idea, our focus is the GPB2 algorithm, which maintains a mixture of $M^2$ Gaussians over time and can be reformulated to include smoothing as well as filtering.

GPB2 is closely related to the Viterbi approximation of Section 3.1. Instead of picking the most likely previous switching state $j$, we collapse the S Gaussians (one for each possible value of $j$) down into a single Gaussian. Namely, the state at time $t$ is obtained as $\hat{x}_{t|t,i} = \sum_j \hat{x}_{t|t,i,j} Pr(s_{t-1} = j | s_t = i, \mathcal{Y}_t)$.

Smoothing in GPB2 is unfortunately a more involved process that includes several additional approximations. Details of this can be found in [12]. Effectively, an RTS smoother can be constructed when an assumption is made that decouples the MC model from the LDS when smoothing the MC states. Together with filtering this results in the following GPB2 algorithm pseudo code

```
Initialize LDS state estimates $\hat{x}_{0|-1,i}$ and $\Sigma_{0|-1,i}$;
Initialize $Pr(s_0 = i| - 1) = \pi(i)$;
for $t = 1 : T - 1$
    for $i = 1 : S$
        for $j = 1 : S$
            Predict and filter LDS state estimates $\hat{x}_{t|t,i,j}, \Sigma_{t|t,i,j}$;
            Find switching state distributions $Pr(s_t = i|\mathcal{Y}_t), Pr(s_{t-1} = j|s_t = i, \mathcal{Y}_t)$;
            Collapse $\hat{x}_{t|t,i,j}, \Sigma_{t|t,i,j}$ to $\hat{x}_{t|t,i}, \Sigma_{t|t,i}$;
        end
        Collapse $\hat{x}_{t|t,i}$ and $\Sigma_{t|t,i}$ to $\hat{x}_{t|t}$ and $\Sigma_{t|t}$;
    end
end
Do GPB2 smoothing;
```

The inference process of GPB2 is more involved than those of the Viterbi or the variational approximation. Unlike Viterbi, GPB2 provides soft estimates of switching states at each time $t$. Like Viterbi GPB2 is a local approximation scheme and as such does not guarantee global optimality inherent in the variational approximation. Some recent work (see [3]) on this type of local approximation in general DBNs has emerged that provides conditions for it to be globally optimal.

## 4  Previous Work

SLDS models and their equivalents have been studied in statistics, time-series modeling, and target tracking since early 1970's. See [13, 12] for a review. Ghahramani [6] introduced a DBN-framework for learning and approximate inference in one class of SLDS models. His underlying model differs from ours in assuming the presence of $S$ independent, white noise-driven LDSs whose measurements are selected by the Markov switching process. A switching framework for particle filters applied to dynamics learning is described in [2]. Manifold learning [7] is another approach to constraining the set of allowable trajectories within a high dimensional state space. An HMM-based approach is described in [4].

## 5  Experimental Results

The data set for our experiments is a corpus of 18 sequences of six individuals performing walking and jogging. Each sequence was approximately 50 frames in duration. All of the motion was fronto-parallel (i.e. occured in a plane that was parallel to the camera plane, as in Figure 2(c).) This simplifies data acquisition and kinematic modeling, while self-occlusions and cluttered backgrounds make the tracking problem non-trivial. Our kinematic model had eight DOF's, corresponding to rotations at the knees, hip, and neck (and ignoring the arms). The link lengths were adjusted manually for each person.

The first task we addressed was learning HMM and SLDS models for walking and running. Each of the two motion types were modeled as one, two, or four-state HMM and SLDS models and then combined into a single complex jog-walk model. In addition, each SLDS motion model was assumed to be of either the first or the second order [2]. Hence, a total of three models (HMM, first order SLDS, and second order SLDS) were considered for each cardinality (one, two, or four) of the switching state.

HMM models were initially assumed to be fully connected. Their parameters were then learned using the standard EM learning, initialized by k-means clustering. Learned HMM models were used to initialize the switching state segmentations for the SLDS models. The SLDS model parameters $(A, Q, R, x_0, \Pi, \pi_0)$ were then reestimated using EM. The inference) in SLDS learning was accomplished using the three approximated methods outlined in Section 3: Viterbi, GPB2, and variational inference.

Results of SLDS learning using either of the three approximate inference methods did not produce significantly different models. This can be explained by the fact that initial segmentations using the HMM and the initial SLDS parameters were all very close to a

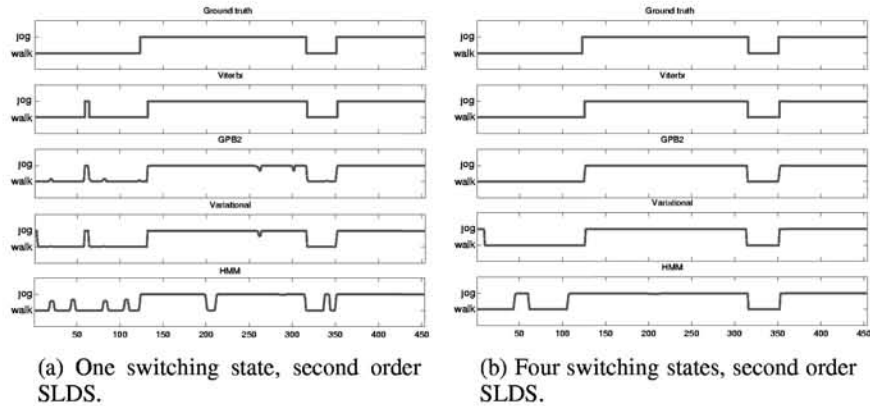

(a) One switching state, second order SLDS.

(b) Four switching states, second order SLDS.

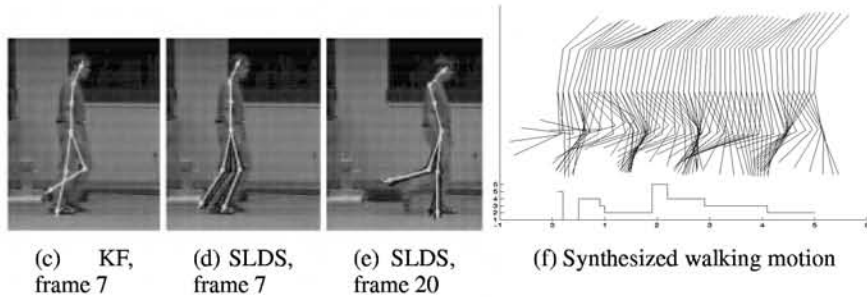

(c)   KF,   (d) SLDS,   (e) SLDS,   (f) Synthesized walking motion
frame 7      frame 7       frame 20

Figure 2: (a)-(d) show an example of classification results on mixed walk-jog sequences using models of different order. (e)-(g) compare constant velocity and SLDS trackers, and (h) shows motion synthesis.

locally optimal solution and all three inference schemes indeed converged to the same or similar posteriors.

We next addressed the classification of unknown motion sequences in order to test the relative performance of inference in HMM and SLDS. Test sequences of walking and jogging motion were selected randomly and spliced together using B-spline smoothing. Segmentation of the resulting sequences into "walk" and "jog" regimes was accomplished using Viterbi inference in the HMM model and approximate Viterbi, GPB2, and variational inference under the SLDS model. Estimates of "best" switching states $Pr(s_t)$ indicated which of the two models were considered to be the source of the corresponding motion segment.

Figure 2(a)-(b) shows results for two representative combinations of switching state and linear model orders. In Figure 2(a), the top graph depicts the true sequence of jog-walk motions, followed by Viterbi, GPB2, variational, and HMM classifications. Each motion type (jog and walk) is modeled using one switching state and a second order LDS. Figure 2(b) shows the result when the switching state is increased to four.

The accuracy of classification increases with the order of the switching states and the LDS model order. More interesting, however, is that the HMM model consistently yields lower segmentation accuracy then all of the SLDS inference schemes. This is not surprising since the HMM model does not impose continuity across time in the plant state space ($x$), which does indeed exist in a natural figure motion (joint angles evolve continuously in time.) Quantitatively, the three SLDS inference schemes produce very similar results. Qualitatively, GPB2 produces "soft" state estimates, while the Viterbi scheme does not. Variational is somewhere in-between. In terms of computational complexity, Viterbi seems

to be the clear winner.

Our next experiment addressed the use of learned dynamic models in visual tracking. The primary difficulty in visual tracking is that joint angle measurements are not readily available from a sequence of image intensities. We use image templates for each link in the figure model, initialized from the first video frame, to track the figure through template registration [11]. A conventional extended Kalman filter using a constant velocity dynamic model performs poorly on simple walking motion, due to pixel noise and self-occlusions, and fails by frame 7 as shown in Figure 2(c). We employ approximate Viterbi inference in SLDS as a multi-hypothesis predictor that initializes multiple local template searches in the image space. From the $S^2$ multiple hypotheses $\hat{x}_{t|t-1,i,j}$ at each time step, we pick the best $S$ hypothesis with the smallest switching cost, as determined by Equation 2. Figure 2(d)-2(e) show the superior performance of the SLDS tracker on the same image sequence. The tracker is well-aligned at frame 7 and only starts to drift off by frame 20. This is not terribly surprising since the SLDS tracker has effectively S (extended) Kalman filters, but it is an encouraging result.

The final experiment simulated walking motion by sampling from a learned SLDS walking model. A stick figure animation obtained by superimposing 50 frames of walking is shown in Figure 2(f). The discrete states used to generate the motion are plotted at the bottom of the figure. The synthesized walk becomes less realistic as the simulation time progresses, due to the lack of global constraints on the trajectories.

## 6   Conclusions

Dynamic models for human motion can be learned within a Switching Linear Dynamic System (SLDS) framework. We have derived three approximate inference algorithms for SLDS: Viterbi, GPB2, and variational. Our variational algorithm is novel in the SLDS domain. We show that SLDS classification performance is superior to that of HMMs. We demonstrate that a tracker based on SLDS is more effective than a conventional Extended Kalman Filter. We show synthesis of natural walking motion by sampling. In future work we will build more complex motion models using a much larger motion capture dataset, which we are currently building. We will also extend the SLDS tracker to more complex measurement models and complex discrete state processes (see [10] for a recent approach).

## Footnotes

[1]SLDS models are sometimes referred to as jump-linear or conditional Gaussian models, and have been studied in the controls and econometrics literatures.

[2]Second order SLDS models imply $x_t = A_1(s_t)x_{t-1} + A_2(s_t)x_{t-2}$.

## References

[1]  Bar-Shalom and Li, *Estimation and tracking: principles, techniques, and software.* 1998.

[2]  A. Blake, B. North, and M. Isard, "Learning multi-class dynamics," in *NIPS '98*, 1998.

[3]  X. Boyen, N. Firedman, and D. Koller, "Discovering the hidden structure of complex dynamic systems," in *Proc. Uncertainty in Artificial Intelligence*, 1999.

[4]  M. Brand, "An entropic estimator for structure discovery," in *NIPS '98*, 1998.

[5]  C. Bregler, "Learning and recognizing human dynamics in video sequences," in *Proc. Int'l Conf. Computer Vision and Pattern Recognition (CVPR)*, 1997.

[6]  Z. Ghahramani and G. E. Hinton, "Switching state-space models." 1998.

[7]  N. Howe, M. Leventon, and W. Freeman, "Bayesian reconstruction of 3d human motion from single-camera video," in *NIPS'99*, 1999.

[8]  M. I. Jordan, Z. Ghahramani, T. S. Jaakkola, and L. K. Saul, "An introduction to variational methods for graphical models," in *Learning in graphical models*, 1998.

[9]  C.-J. Kim, "Dynamic linear models with markov-switching," *J. Econometrics*, vol. 60, 1994.

[10]  U. Lerner, R. Parr, D. Koller, and G. Biswas, "Bayesian fault detection and diagnosis in dynamic systems," in *Proc. AAAI*, (Austin, TX), 2000.

[11]  D. Morris and J. Rehg, "Singularity analysis for articulated object tracking," in *CVPR*, 1998.

[12]  K. P. Murphy, "Learning switching kalman-filter models," TR 98-10, Compaq CRL., 1998.

[13]  V. Pavlović, J. M. Rehg, T.-J. Cham, and K. P. Murphy, "A dynamic bayesian network approach to figure tracking using learned dynamic models," in *Proc. Intl. Conf. Computer Vision*, 1999.
